# Integration of Visual and Somatosensory Information for Preshaping Hand in Grasping Movements

**Yoji Uno**
ATR Human Information Processing
Research Laboratories
2-2 Hikaridai, Seika-cho, Soraku-gun,
Kyoto 619-02, Japan

**Naohiro Fukumura***
Faculty of Engineering
University of Tokyo
7-3-1 Hongo, Bunkyo-ku,
Tokyo 113, Japan

**Ryoji Suzuki**
Faculty of Engineering
University of Tokyo
7-3-1 Hongo, Bunkyo-ku,
Tokyo 113, Japan

**Mitsuo Kawato**
ATR Human Information Processing
Research Laboratories
2-2 Hikaridai, Seika-cho, Soraku-gun,
Kyoto 619-02, Japan

## Abstract

The primate brain must solve two important problems in grasping movements. The first problem concerns the recognition of grasped objects: specifically, how does the brain integrate visual and motor information on a grasped object? The second problem concerns hand shape planning: specifically, how does the brain design the hand configuration suited to the shape of the object and the manipulation task? A neural network model that solves these problems has been developed. The operations of the network are divided into a *learning phase* and an *optimization phase*. In the learning phase, internal representations, which depend on the grasped objects and the task, are acquired by integrating visual and somatosensory information. In the optimization phase, the most suitable hand shape for grasping an object is determined by using a relaxation computation of the network.

*Present Address: Parallel Distributed Processing Research Dept., Sony Corporation, 6-7-35 Kitashinagawa, Shinagawa-ku, Tokyo 141, Japan

## 1  INTRODUCTION

It has previously been established that, while reaching out to grasp an object, the human hand preshapes according to the shape of the object and the planned manipulation (Jeannerod, 1984; Arbib et al., 1985). The preshaping of the human hand suggests that prior to grasping an object the 3-dimensional form of the object is recognized and the most suitable hand configuration is preset depending on the manipulation task.

It is supposed that the human recognizes objects using not only visual information but also somatosensory information when the hand grasps them. Visual information is made from the 2-dimensional image in the visual system of the brain. Somatosensory information is closely related to motor information, because it depends on the prehensile hand shape (i.e., finger configuration). We hypothesize that an internal representation of a grasped object is formed in the brain by integrating visual and somatosensory information. Some physiological studies support our hypothesis. For example, Taira et al. (1990) found that the activity of *hand-movement-related neurons* in the posterior parietal association cortex were highly selective to the shape and/or the orientation of manipulated switches.

How can the neural network integrate different kinds of information? Merely uniting visual image with somatosensory information does not lead to any interesting representation. Our basic idea is that information compression is applied to integrating different kinds of information. It is useful to extract the essential information by compressing the visual and somatosensory information.

Irie & Kawato (1991) pointed out that multi-layered perceptrons have the ability to extract features from the input signals by compressing the information from input signals. Katayama & Kawato (1990) proposed a learning schema in which an internal representation of the grasped object was acquired using information compression. Developing the schema of Katayama et al., we have devised a neural network model for recognizing objects and planning hand shapes (e.g., Fukumura et al. 1991). This neural network consists of five layers of neurons with only forward connections as shown in Figure 1. The input layer (1st layer) and the output layer (5th layer) of the network have the same structure. There are fewer neurons in the 3rd layer than in the 1st and 5th layers. The operations of the network are divided into the *learning phase*, which is discussed in section 2 and the *optimization phase*, which is discussed in section 3.

## 2  INTEGRATION OF VISUAL AND SOMATOSENSORY INFORMATION USING NETWORK LEARNING

In the learning phase, the neural network learns the relation between the visual information (i.e.,visual image) and the somatosensory information which, in this paper, is regarded as information on the prehensile hand configuration (i.e., finger configuration).

Both vector x representing the visual image of an object and vector y representing the prehensile hand configuration to grasp it are fed into the 1st layer (the input layer). The synaptic weights of the network are repeatedly adjusted so that the 5th layer outputs the same vectors x and y as are fed into the 1st layer. In other words, the network comes to realize the **identity map** between the 1st layer and the 5th layer through a learning process. The most important point of the neural network model is that the number of neurons in the 3rd layer is smaller than the number of neurons in the 1st layer (which is equal to the number of neurons in the 5th layer). Therefore, the information from x and y is compressed between

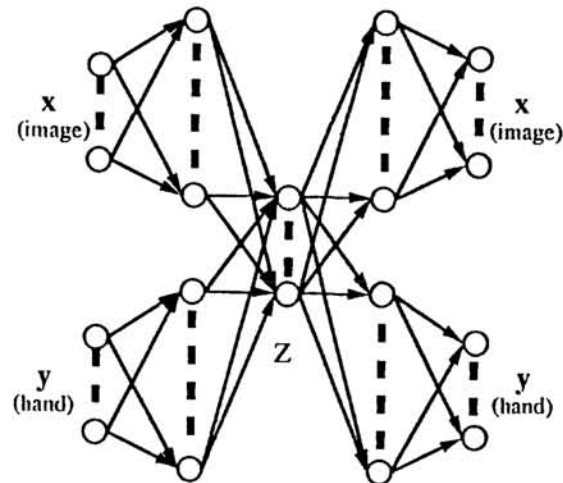

Figure 1: A neural network model for integrating visual image **x** and prehensile hand configuration **y**. The internal representation **z** of a grasped object is acquired in the third layer.

the 1st layer and the 3rd layer, and restored between the 3rd layer and the 5th layer. Once the network learning process is complete, visual image **x** and prehensile hand configuration **y** are integrated in the network. Consequently, the internal representation **z** of the grasped object, which should include enough information to reproduce **x** and **y**, is formed in the 3rd layer.

Prehensile hand configuration in grasping movements were measured and the learning of the network was simulated by a computer. In behavioral experiments, three kinds of wooden objects were prepared: five circular cylinders whose diameters were 3 cm, 4 cm, 5 cm, 6 cm and 7 cm; four quadrangular prisms whose side lengths were 3 cm, 4 cm, 5 cm and 6 cm; and three spheres whose diameters were 3 cm, 4 cm and 5 cm. Data input to the network was comprised of visual image **x** and prehensile hand configuration **y**.

Visual images of objects are formed through complicated processes in the visual system of the brain. For simplicity, however, projections of objects onto a side plane and/or a bottom plane were used instead of real visual images. The area of each pixel of the projected image was fed into the network as an element of visual image **x**. A $DataGlove^{TM} (VPL)$ was used to measure finger configurations in grasping movements. We attached sixteen optical fibers, whose outputs were roughly inversely proportional to finger joint-angles, to the DataGlove. The subject was instructed to grasp the objects on the table tightly with the palm and all the fingers. The subject grasped twelve objects thirty times each, which produced 360 prehensile patterns for use as training data for network learning.

In the computer simulation, six neurons were set in the 3rd layer. The back-propagation learning method was applied in order to modify the synaptic weights in the network. Figure 2 shows the activity of neurons in the 3rd layer after the learning had sufficiently been performed. Some interesting features of the internal representations were found in Figure 2. The first is that the level of neuron activity in the 3rd layer increased monotonically as the size of the object increased. The second is that, except for the magnitude, the neuron activation patterns for the same kinds of objects were almost the same. Furthermore, the activation patterns were similar for circular cylinders and quadrangular prisms, but were quite different for spheres. In other words, similar representations were acquired for similarly shaped objects. We concluded that the internal represenations were formed in the 3rd

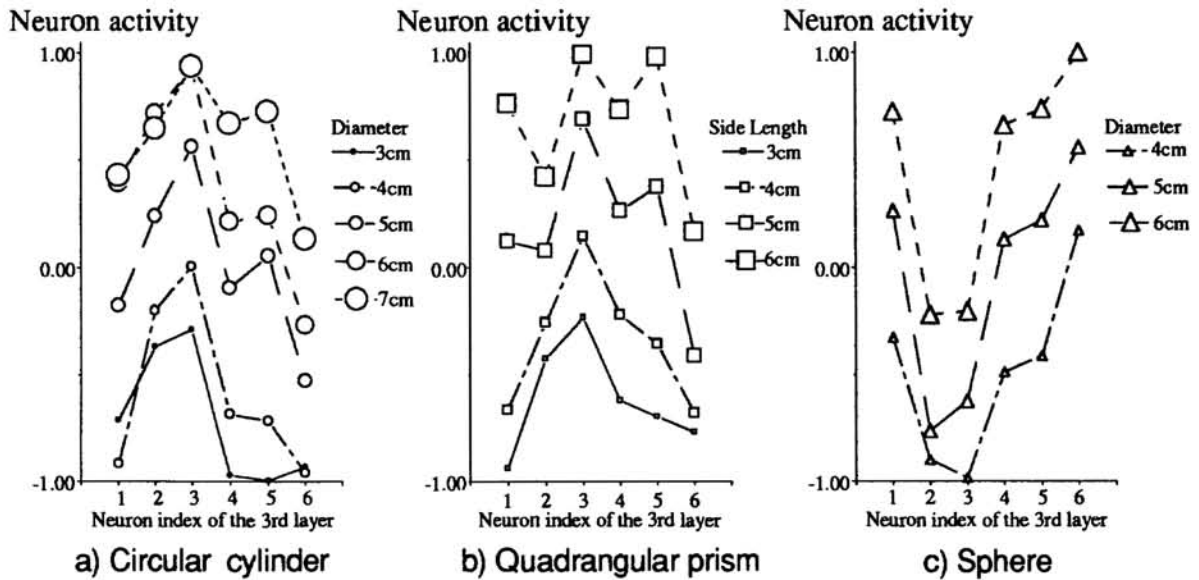

a) Circular cylinder        b) Quadrangular prism        c) Sphere

Figure 2: Internal representations of grasped objects. Graph **a)** shows the neuron activation patterns for five circular cylinders whose diameters were 3 cm, 4 cm, 5 cm, 6 cm and 7 cm. Graph **b)** shows the neuron activation patterns for four quadrangular prisms whose side lengths were 3 cm, 4 cm, 5 cm and 6 cm. Finally, Graph **c)** shows the neuron activation patterns for three spheres whose diameters were 3 cm, 4 cm and 5 cm. The abscissa represents the index of the six neurons in the 3rd layer, while the ordinate represents their activity. These values were normalized from -1 to +1.

layer and changed topologically according to the shapes and sizes of the grasped objects.

## 3   DESIGN OF PREHENSILE HAND SHAPES

The neural network that has completed the learning can design hand shapes to grasp any objects in the optimization phase. Determining prehensile hand shape (i.e., finger configuration) is an ill-posed problem, because there are many ways to grasp any given object. In other words, prehensile hand configuration cannot be determined uniquely for any one object. In order to solve this indeterminacy, a criterion, a measure of performance for any possible prehensile configuration is introduced.

The criterion should normally be defined based on the dynamics of the human hand and the manipulation task. However, for simplicity, the criterion is defined based only on the static configuration of the fingers, which is represented by vector **y**. We assumed that the central nervous system adopts a stable hand configuration to grasp an object, which corresponds to flexing the fingers as much as possible. The output of the DataGlove sensor decreases as finger flexion increases. Therefore, the criterion $C_1(\mathbf{y})$ is defined as follows:

$$C_1(\mathbf{y}) = \frac{1}{2} \sum_i y_i^2, \tag{1}$$

where $y_i$ represents the $i$th output of the sixteen DataGlove sensors. Minimizing the criterion $C_1(\mathbf{y})$ requires as much finger flexing as possible.

Finding values of $y_i$ ($i = 1, 2, \ldots, 16$) so as to minimize $C_1(\mathbf{y})$ is an optimization problem

with constraints. In the optimization phase, the neural network can solve this optimization problem using a relaxation computation as follows. When an object is specified, the visual image $\mathbf{x}^*$ of the object is input to the 1st layer as an input signal and given to the 5th layer as a reference signal. We call neurons in the 1st and the 5th layers which represent visual image $\mathbf{x}$ *image neurons*, and call neurons in the 1st and the 5th layers which represent finger configuration $\mathbf{y}$ *hand neurons*. Let us define the following energy function of the network.

$$E(\mathbf{y}) = \frac{1}{2}\sum_i (x_i^* - x_i')^2 + \frac{1}{2}\sum_j (y_j - y_j')^2 + \lambda \cdot \frac{1}{2}\sum_j y_j^2. \qquad (2)$$

Here, $x_i^*$ is the $i$th element of the image $\mathbf{x}^*$ which is fed into the $i$th image neuron in the 1st layer, and $x_i'$ is the output of the $i$th image neuron in the 5th layer. $y_j$ is the activity of the $j$th hand neuron in the 1st layer, and $y_j'$ is the output of the $j$th hand neuron in the 5th layer. $\lambda$ is a positive regularization parameter which decreases gradually during the relaxation computation. The first term and the second term of equation (2) require that the network realizes the identity map between the input layer and the output layer as well as in the learning phase. This requirement guarantees that a hand whose configuration is specified by vector $\mathbf{y}$ can grasp an object whose visual image is $\mathbf{x}^*$. The third term of equation (2) represents the criterion $C_1(\mathbf{y})$. In the optimization phase, the values of the synaptic weights are fixed. Instead, the hand neuron changes its state autonomously while obeying the following differential equation:

$$c\frac{dy_k}{ds} = -\frac{\partial E}{\partial y_k}, \quad k = 1, 2, \ldots, 16. \qquad (3)$$

Here, $s$ is the relaxation time required for the state change of the hand neuron, and $c$ is a positive time constant. The right-hand side of equation (3) can be transformed as follows:

$$-\frac{\partial E}{\partial y_k} = \sum_i (x_i^* - x_i')\frac{\partial x_i'}{\partial y_k} + \sum_j (y_j - y_j')\frac{\partial y_j'}{\partial y_k} + (y_k - y_k')\left(\frac{\partial y_k'}{\partial y_k} - 1\right) - \lambda y_k. \qquad (4)$$

It is straightforward to show that the first three terms of equation (4) are the error signals at the $k$th hand neuron, which can be calculated backward from the output layer to the input layer. The fourth term of equation (4) is a suppressive signal which is given to the hand neuron by itself. When the state of the hand neuron obeys the differential equation (3), the time change $E$ can be expressed as :

$$\frac{dE}{ds} = \sum_k \frac{dy_k}{ds}\frac{\partial E}{\partial y_k} = -c\sum_k \left(\frac{dy_k}{ds}\right)^2 \leq 0. \qquad (5)$$

Therefore, the energy function E always decreases and the network comes to the equilibrium state that is the (local) minimum energy state. The outputs of the hand neurons in the equilibrium state represent the solution of the optimization problem which corresponds to the most suitable finger configuration.

The relaxation computation of the neural network was simulated. For example, when given the image of a circular cylinder whose diameter was 5 cm, the prehensile finger configuration was computed. After a hundred-thousand iterations for the relaxation computation, we had the results shown in Figure 3. The left sied shows the hand shape that had the minimum value of the criterion of all the training data recorded when the subject grasped a circular cylinder whose diameter was 5 cm. The right side shows the hand shape produced by relaxation computation. These two hand shapes were very similar, which indicated that the network reproduced hand shape by using relaxation computation.

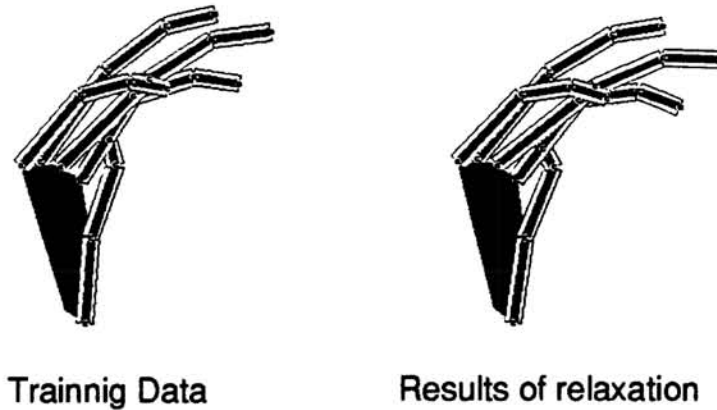

Trainnig Data          Results of relaxation

Figure 3: Prehesile hand shapes for grasping a circular cylinder whose diameter was 5cm.

## 4   VARIOUS TYPES OF PREHENSIONS

In the sections above, the subject was instructed to grasp objects using only one type of prehension. It is, however, thought that a human chooses different types of prehensions depending on the manipulation tasks. In order to investigate the dependence of the internal representation on the type of prehension, the second behavioral experiment was conducted. In this experiment, five circular cylinders and three spheres which were the same size as those in the first experiment were prepared. The subject was first instructed to grasp the objects tightly with his palm and all of his fingers, and then to grasp the same objects with only his fingertips. Iberall et al.(1988) referred to the first prehension and the second prehension as *palm opposition* and *pad opposition*, respectively. The subject grasped eight objects in two different types of prehensions twenty times each, which produced 320 prehensile patterns. Four neurons were set in the 3rd layer of the network and the network learning was simulated using these prehensile patterns as training data. Figure 4 shows the neuronal activation patterns formed in the 3rd layer after the network learning. Even if the grasped objects were the same, the neuron activation pattern for palm opposition was quite different from that for palm opposition.

The neural network can reproduce different prehension, by introducing different criteria. $C_1(\mathbf{y})$ is definded corresponding to palm opposition. Furthermore, we define another criterion $C_2(\mathbf{y})$, corresponding to pad opposition.

$$C_2(\mathbf{y}) = \sum^{i \in MP, CM} y_i^2 + \sum^{i \in IP} (1.0 - y_j)^2. \tag{6}$$

Minimizing the criterion $C_2(\mathbf{y})$ demands that the MP joints (metacarpophalangeal joints) of the four fingers and the CM joint (carpometacarpal joint) of the thumb be flexed as much as possible and that the IP joints (interphalangeal joints) of all five fingers be stretched as much as possible. The relaxation computation of the neural network was simulated, when given the image of a sphere whose diameter was 5 cm. The results of the relaxation computation are shown in Figure 5. Adopting the different criteria, the neural network reproduced different prehensile hand configurations which corresponded to a) palm opposition and b) pad opposition.

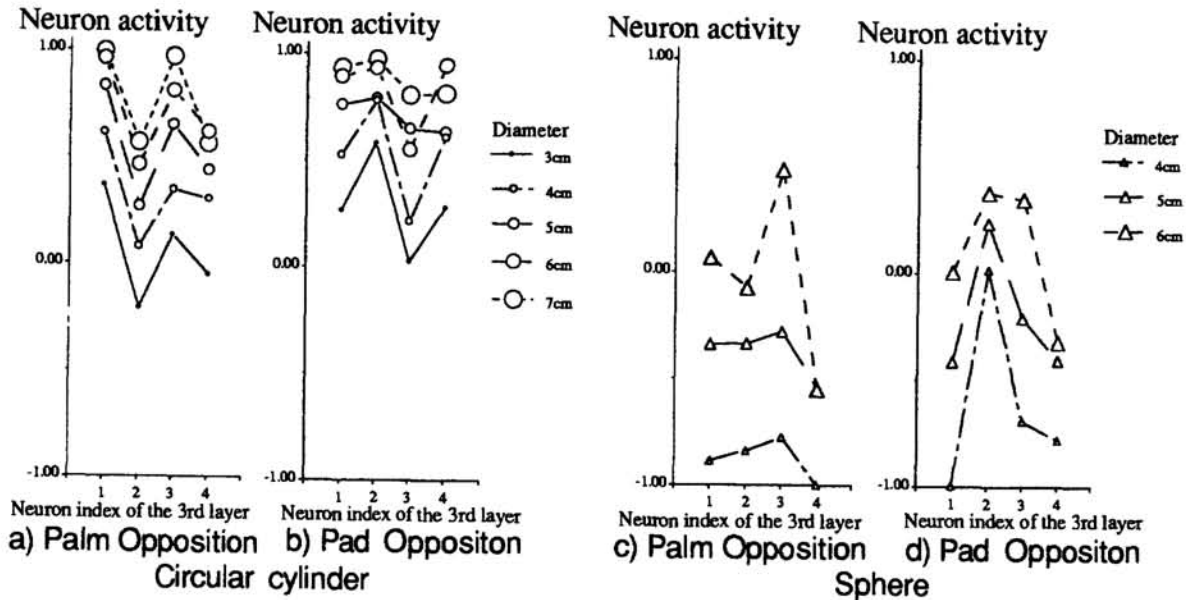

Figure 4: Internal representations of grasped objects formed in the 3rd layer of the network. Graphs **a**), **b**), **c**) and **d**) show the activation patterns of neurons for palm oppositions when grasping 5 circular cylinders, for pad oppositions when grasping 5 circular cylinders, for palm oppositions when grasping 3 spheres and for pad oppositions when grasping 3 spheres, respectively. See Figure 2 legend for description.

## 5   DISCUSSION

In view of the function of neurons in the posterior parietal association cortex, we have devised a neural network model for integrating visual and motor information. The proposed neural network model is an active sensing model, as it learns only when an object is successfully grasped. In this paper, tactile information is not treated, as the materials of the grasped objects are not considered for simplicity. We know that tactile information plays an important role in the recognition of grasped objects. The neural network model shown in Figure 1 can easily be developed so as to integrate visual, motor and tactile information. However, it is not clear how the internal representations of grasped objects is changed by adding tactile information.

The critical problem in our neural network model is how many neurons should be set in the 3rd layer to represent the shapes of grasped objects. If there are too few neurons in the 3rd layer, the 3rd layer cannot represent enough information to restor x and y between the 3rd layer and the 5th layer; that is, the network cannot learn to realize the identity map between the input layer and the output layer. If there are too many neurons in the 3rd layer, the network cannot obtain useful representations of the grasped objects in the 3rd layer and the relaxation computation sometimes fails. In the present stage, we have no method to decide an adequate number of neurons for the 3rd layer. This is an important task for the future.

**Acknowledgements**
The main part of this study was done while the first author (Y.U.) was working at University of Tokyo. Y. Uno, N. Fukumura and R. Suzuki was supported by Japanese Ministry of

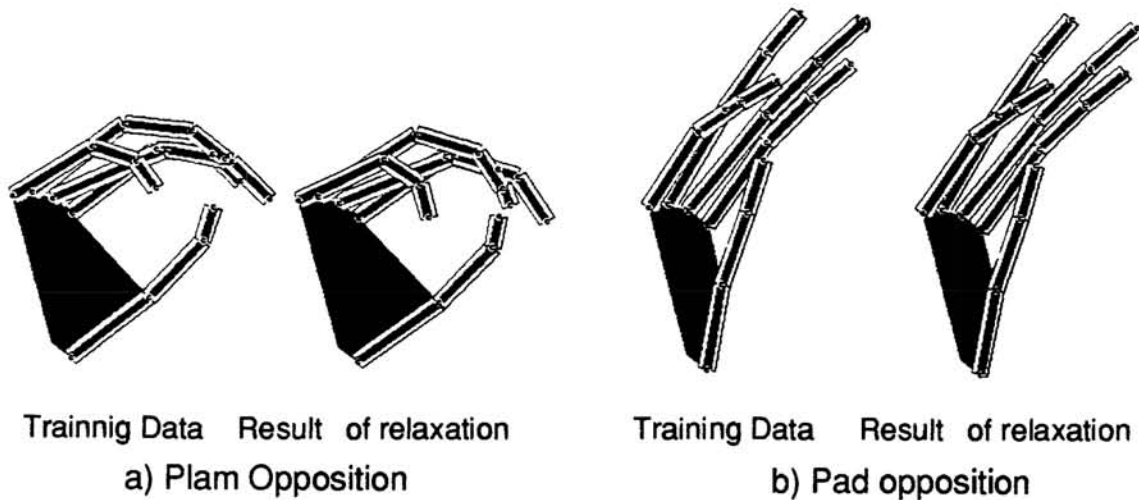

Trainnig Data    Result  of relaxation        Training Data    Result  of relaxation

a) Plam Opposition                          b) Pad opposition

Figure 5: Prehensile hand configuration **a**) for palm opposition and prehensile hand configuration **b**) for pad opposition when grasping a sphere whose diameter was 5 cm. The left sides show the hand shapes with the minimum values of the criterions for all training data recorded when the subject grasped a sphere whose diameter was 5 cm. The right sides show the hand shapes made by the relaxation computation.

Education, Science and Culture Grants, No.03251102 and No.03650338. M. Kawato was supported by Human Frontier Science Project Grant.

**References**

M. Jeannerod. (1984) The timing of natural prehension movements, *J. Motor Behavior*, **16**: 235-254.

M.A. Arbib, T. Iberall and D. Lyons. (1985) Coordinated control programs for movements of the hand. *Hand Function and the Neocortex. Experimental Brain Research*, suppl.**10**, 111-129.

N. Fukumura, Y. Uno, R. Suzuki and K. Kawato (1991) A neural network model which recognizes shape of a grasped object and decides hand configuration. *Japan IEICE Technical Report*, NC90-104: 213-218 (in Japanese).

Katayama and M. Kawato (1990) Neural network model integrating visual and somatic information. *J. Robotics Society of Japan*, **8**: 117-125 (in Japanese).

T. Iberall (1998) A neural network for planning hand shapes in human prehension. *proc. Automation and controls Conf.*: 2288-2293.

B. Irie and Kawato (1991) "Acquisition of Internal Representation by Multilayered Perceptrons." *Electronics and Communications in Japan*, Part 3, **74**: 112-118.

M. Taira, S. Mine, A.P. Georgopoulos, A. Murata and S. Sakata. (1990) Parietal cortex neurons of the monkey related to the visual guidance of hand movement. *Exp. Brain Res.*, **83**: 29-36.
